# Comparison of Human and Machine Word Recognition

**M. Schenkel**
Dept of Electrical Eng.
University of Sydney
Sydney, NSW 2006, Australia
schenkel@sedal.usyd.edu.au

**C. Latimer**
Dept of Psychology
University of Sydney
Sydney, NSW 2006, Australia

**M. Jabri**
Dept of Electrical Eng.
University of Sydney
Sydney, NSW 2006, Australia
marwan@sedal.usyd.edu.au

## Abstract

We present a study which is concerned with word recognition rates for heavily degraded documents. We compare human with machine reading capabilities in a series of experiments, which explores the interaction of word/non-word recognition, word frequency and legality of non-words with degradation level. We also study the influence of character segmentation, and compare human performance with that of our artificial neural network model for reading. We found that the proposed computer model uses word context as efficiently as humans, but performs slightly worse on the pure character recognition task.

## 1 Introduction

Optical Character Recognition (OCR) of machine-print document images has matured considerably during the last decade. Recognition rates as high as 99.5% have been reported on good quality documents. However, for lower image resolutions (200 DpI and below), noisy images, images with blur or skew, the recognition rate declines considerably. In bad quality documents, character segmentation is as big a problem as the actual character recognition. In many cases, characters tend either to merge with neighbouring characters (dark documents) or to break into several pieces (light documents) or both. We have developed a reading system based on a combination of neural networks and **hidden Markov models** (HMM), specifically for low resolution and degraded documents.

To assess the limits of the system and to see where possible improvements are still to be

expected, an obvious comparison is between its performance and that of the best reading system known, the human reader. It has been argued, that humans use an extremely wide range of context information, such as current topics, syntax and semantic analysis in addition to simple lexical knowledge during reading. Such higher level context is very hard to model and we decided to run a first comparison on a word recognition task, excluding any context beyond word knowledge.

The main questions asked for this study are: how does human performance compare with our system when it comes to pure character recognition (no context at all) of bad quality documents? How do they compare when word context can be used? Does character segmentation information help in reading?

## 2 Data Preparation

We created as stimuli 36 data sets, each containing 144 character strings, 72 words and 72 non-words, all lower case. The data sets were generated from 6 original sets, each containing 144 unique words/non-words. For each original set we used three ways to divide the words into the different degradation levels such that each word appears once in each degradation level. We also had two ways to pick segmented/non-segmented so that each word is presented once segmented and once non-segmented. This counterbalancing creates the 36 sets out of the six original ones. The order of presentation within a test set was randomized with respect to degradation, segmentation and lexical status.

All character strings were printed in 'times roman 10 pt' font. Degradation was achieved by photocopying and faxing the printed document before scanning it at 200DpI. Care was taken to randomize the print position of the words such that as few systematic degradation differences as possible were introduced.

Words were picked from a dictionary of the 44,000 most frequent words in the 'Sydney Morning Herald'. The length of the words was restricted to be between 5 and 9 characters. They were divided in a 3x3x2 mixed factorial model containing 3 word-frequency groups, 3 stimulus degradation levels and visually segmented/non-segmented words. The three word-frequency groups were: 1 to 10 occurences/million (o/m) as low frequency, 11 to 40 o/m as medium frequency and 41 or more o/m as high frequency. Each participant was presented with four examples per stimulus class (e.g. four high frequency words in medium degradation level, not segmented).

The non-words conformed to a 2x3x2 model containing legal/illegal non-words, 3 stimulus degradation levels and visually segmented/non-segmented strings. The illegal non-words (e.g. 'ptvca') were generated by randomly selecting a word length between 5 and 9 characters (using the same word length frequencies as the dictionary has) and then randomly picking characters (using the same character frequencies as the dictionary has) and keeping the unpronounceable sequences. The legal non-words (e.g. 'slunk') were generated by using trigrams (using the dictionary to compute the trigram probabilities) and keeping pronounceable sequences. Six examples per non-word stimulus class were used in each test set. (e.g. six illegal non-words in high degradaton level, segmented).

## 3 Human Reading

There were 36 participants in the study. Participants were students and staff of the University of Sydney, recruited by advertisement and paid for their service. They were all native English speakers, aged between 19 and 52 with no reported uncorrected visual deficits.

The participants viewed the images, one at a time, on a computer monitor and were asked to type in the character string they thought would best fit the image. They had been

instructed that half of the character strings were English words and half non-words, and they were informed about the degradation levels and the segmentation hints. Participants were asked to be as fast and as accurate as possible. After an initial training session of 30 randomly picked character strings not from an independent training set, the participants had a short break and were then presented with the test set, one string at a time. After a Carriage Return was typed, time was recorded and the next word was displayed. Training and testing took about one hour. The words were about 1-1.5cm large on the screen and viewed at a distance of 60cm, which corresponds to a viewing angle of 1°.

# 4  Machine Reading

For the machine reading tests, we used our integrated segmentation/recognition system, using a sliding window technique with a combination of a neural network and an HMM [6]. In the following we describe the basic workings without going into too much detail on the specific algorithms. For more detailed description see [6].

A sliding window approach to word recognition performs no segmentation on the input data of the recognizer. It consists basically of sweeping a window over the input word in small steps. At each step the window is taken to be a tentative character and corresponding character class scores are produced. Segmentation and recognition decisions are then made on the basis of the sequence of character scores produced, possibly taking contextual information into account.

In the **preprocessing** stage we normalize the word to a fixed height. The result is a grey-normalized pixel map of the word. This pixel map is the input to a **neural network** which estimates a *posteriori* probabilities of occurrence for each character given the input in the sliding window whose length corresponds approximately to two characters. We use a space displacement neural network (SDNN) which is a multi-layer feed-forward network with local connections and shared weights, the layers of which perform successively higher-level feature extraction. SDNN's are derived from Time Delay Neural Networks which have been successfully used in speech recognition [2] and handwriting recognition [4, 1]. Thanks to its convolutional structure the computational complexity of the sliding window approach is kept tractable. Only about one eighth of the network connections are reevaluated for each new input window. The outputs of the SDNN are processed by an HMM. In our case the HMM implements character duration models. It tries to align the best scores of the SDNN with the corresponding expected character durations. The Viterbi algorithm is used for this alignment, determining simultaneously the segmentation and the recognition of the word. Finding this state sequence is equivalent to finding the most probable path through the graph which represents the HMM. Normally additive costs are used instead of multiplicative probabilities. The HMM then selects the word causing the smallest costs.

Our best architecture contains 4 convolutional layers with a total of 50,000 parameters [6]. The training set consisted of a subset of 180,000 characters from the SEDAL database, a low resuloution degraded document database which was collected earlier and is independent of any data used in this experiment.

## 4.1  The Dictionary Model

A natural way of including a **dictionary** in this process, is to restrict the solution space of the HMM to words given by the dictionary. Unfortunately this means calculating the cost for each word in the dictionary, which becomes prohibitively slow with increasing dictionary size (we use a combination of available dictionaries with a total size of 98,000 words). We thus chose a two step process for the dictionary search: in a first step a list of the most probable words is generated, using a fast-matcher technique. In the second step the HMM costs are calculated for the words in the proposed list.

To generate the word list, we take the character string as found by the HMM *without* the dictionary and calculate the edit-distance between that string and all the words in the dictionary. The edit-distance measues how many edit operations (insertion, deletion and substitution) are necessary to convert a given input string into a target word [3, 5]. We now select all dictionary words that have the smallest edit-distance to the string recognized without using the dictionary. The composed word list contains on average 10 words, and its length varies considerably depending on the quality of the initial string.

For all words in the word list the HMM cost is now calculated and the word with the smallest cost is the proposed dictionary word. As the calculation of the edit-distance is much faster than the calculation of the HMM costs, the recognition speed is increased substantially.

In a last step the difference in cost between the proposed dictionary word and the initial string is calculated. If this difference is smaller than a threshold, the system will return the dictionary word, otherwise the original string is returned. This allows for the recognition of non-dictionary words. The value for the threshold determines the amount of reliance on the dictionary. A high value will correct most words but will also force non-words to be recognized as words. A low value, on the other hand, leaves the non-words unchanged but doesn't help for words either. Thus the value of the threshold influences the difference between word and non-word recognition. We chose the value such that the over-all error rate is optimized.

## 4.2 The Case of Segmented data

When character segmentation is given, we know how many characters we have and where to look for them. There is no need for an HMM and we just sum up the character probabilities over the x-coordinate in the region corresponding to a segment. This leaves a vector of 26 scores (the whole alphabet) for each character in the input string. With no dictionary constraints, we simply pick the label corresponding to the highest probability for each character. The dictionary is used in the same way, replacing the HMM scores by calculating the word scores directly from the corresponding character probabilities.

# 5 Results

## Recognition Performance

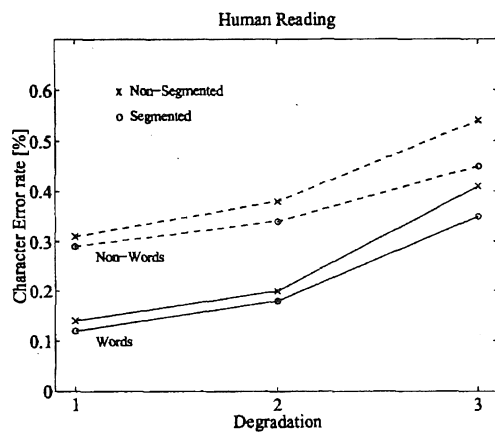 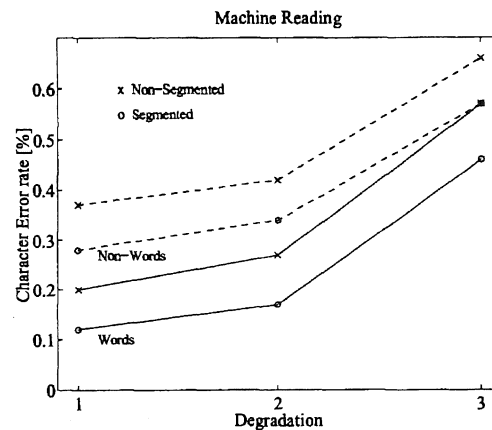

Figure 1: **Human Reading Performance.**    Figure 2: **Machine Reading Performance.**

Figure 1 depicts the recognition results for human readers. All results are per character error rates counted by the edit-distance. All results reported as significant pass an F-test with $p < .01$. As expected there was a significant interaction between error rate and degradation and clearly non-words have higher error rates than words. Also character segmentation has also an influence on the error rate. Segmentation seems to help slightly more for higher degradations.

Figure 2 shows performance of the machine algorithm. Again greater degradation leads to higher error rates and non-words have higher error rates than words. Segmentation hints lead to significantly better recognition for all degradation levels; in fact there is no interaction between degradation and segmentation for the machine algorithm. In general the machine benefited more from segmentation than humans.

One would expect a smaller gain from lexical knowledge for higher error rates (i.e. higher degradation) as in the limit of complete degradation *all* error rates will be 100%. Both humans and machine show this 'closing of the gap .

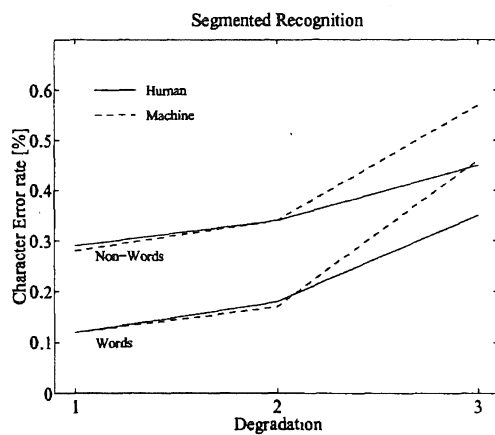 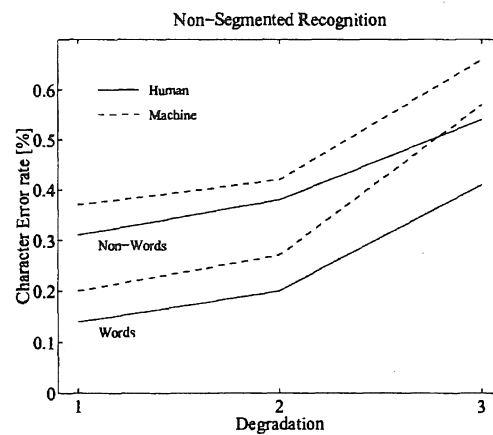

Figure 3: **Segmented Data.**          Figure 4: **Non-Segmented Data.**

More interesting is the direct comparison between the error rates for humans and machine as shown in figure 3 and figure 4. The difference for non-words reflects the difference in ability to recognize the geometrical shape of characters without context. For degradation levels 1 and 2, the machine has the same reading abilities as humans for segmented data and looses only about 7% in the non-segmented case. For degradation level 3, the machine clearly performs worse than human readers.

The difference between word and non-word error rates reflects the ability of the participant to use lexical knowledge. Note that the task contains word/non-word discrimination as well as recognition. It is striking how similar the behaviour for humans and machine is for degradation levels 1 and 2.

**Timing Results**

Figure 5 shows the word entry times for humans. As the main goal was to compare recognition rates, we did not emphasize entry speed when instructing the participants. However, we recorded the word entry time for each word (which includes inspection time and typing). When analysing the timimg data the only interest was in relative difference between word groups. Times were therefore converted for each participant into a z-score (zero mean with a standard deviation of one) and statistics were made over the z-scores of all participants.

Non-words generally took longer to recognize than words and segmented data took longer

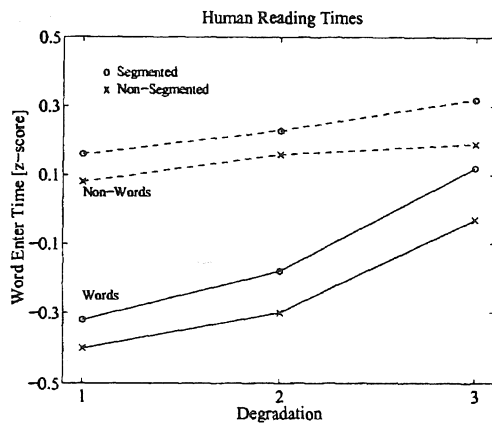

Figure 5: **Human Reading Times.**

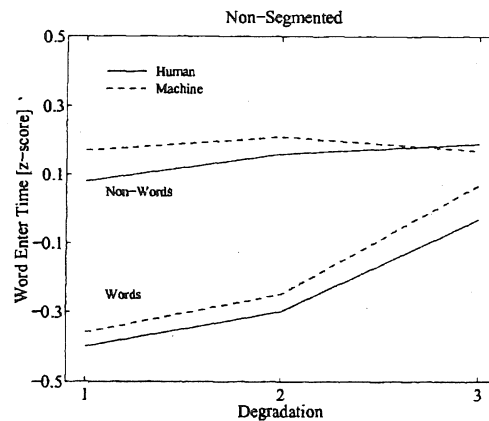

Figure 6: **Non-Segmented Reading Times.**

to recognize than non-segmented for humans which we believe stems from participants not being used to reading segmented data. When asked, participants reported difficulties in using the segmentation lines. Interestingly this segmentation effect is significant only for words but not for non-words.

As predicted there is also an interaction between time and degradation. Greater degradations take longer to recognize. Again, the degradation effect for time is only significant for words but barely for non-words.

Our machine reading algorithm behaves differently in segmented and non-segmented mode with respect to time consumption. In segmented mode, the time for evaluating the word list in our system is very short compared to the base recognition time, as there is no HMM involved. Accordingly we found no or very little effects on timing for our system for segmented data. All the timing information for the machine refer to the non-segmented case (see Figure 6).

**Frequency and Legality**

Table 5 shows word frequencies, legality of non-words and entry-time. Our experiment confirmed the well known frequency and legality effect for humans in recognition rate as well as time and respectively for frequency. The only exception is that there is no difference in error rate for middle and low frequency words.

The machine shows (understandably) no frequency effect in *error rate* or time, as all lexical words had the same prior probability. Interestingly even when using the correct prior probabilities we could not produce a strong word frequency effect for the machine. Also no legality effect was observed for the *error rate*. One way to incorporate legality effects would be the use of Markov chains such as n-grams.

Note however, how the recognition time for non-words is higher than for words and the legality effect for the recognition time. Recognition times for our system in non-segmented mode depend mainly on the time it takes to evaluate the word list. Non-words generally produce longer word lists than words, because there are no good quality matches for a non-word in the dictionary (on average a word list length of 8.6 words was found for words and of 14.5 for non-words). Also illegal non-words produce longer word lists than legal ones, again because the match quality for illegal non-words is worse than for legal ones (average length for illegal non-words 15.9 and for legal non-words 13.2). The z-scores for the word list length parallel nicely the recognition time scores.

In segmented mode, the time for evaluating the word list is very short compared to the

base recognition time, as there is no HMM involved. Accordingly we found no or very little effects on timing for our system in the segmented case.

| Error [%] | Humans | | Machine | |
|---|---|---|---|---|
| | Error | z-Time | Error | z-Time |
| Words 41+ | 0.22 | -0.37 | 0.36 | -0.14 |
| Words 11-40 | 0.27 | -0.13 | 0.34 | -0.19 |
| Words 1-10 | 0.26 | -0.06 | 0.33 | -0.22 |
| Legal Non-W. | 0.36 | 0.07 | 0.47 | 0.09 |
| Illegal Non-W. | 0.46 | 0.31 | 0.49 | 0.28 |

Table 1: **Human and Machine Error rates for the different word and non-word classes.** The z-times for the machine are for the non-segmented data only.

# 6   Discussion

The ability to recognize the geometrical shape of characters without the possibility to use any sort of context information is reflected in the error rate of illegal non-words. The difference between the error rate for illegal non-words and the one for words reflects the ability to use lexical knowledge. To our surprise the behavior of humans and machine is very similar for both tasks, indicating a near to optimal machine recognition system. Clearly this does not mean our system is a good model for human reading. Many effects such as semantic and repetition priming are not reproduced and call for a system which is able to build semantic classes and memorize the stimuli presented. Nevertheless, we believe that our experiment validates empirically the verification model we implemented, using real world data.

# Acknowledgments

This research is supported by a grant from the Australian Research Council (grant No A49530190).

# References

[1] I. Guyon, P. Albrecht, Y. Le Cun, J. Denker, and W. Hubbard. Design of a neural network character recognizer for a touch terminal. *Pattern Recognition*, 24(2):105–119, 1991.

[2] K. J. Lang and G. E. Hinton. A Time Delay Neural Network architecture for speech recognition. Technical Report CMU-cs-88-152, Carnegie-Mellon University, Pittsburgh PA, 1988.

[3] V. I. Levenshtein. Binary codes capable of correcting deletions, insertions and reversals. *Soviet Physics-Doklady*, 10(8):707–710, 1966.

[4] O. Matan, C. J. C. Burges, Y. Le Cun, and J. Denker. Multi-digit recognition using a Space Dispacement Neural Network. In J. E. Moody, editor, *Advances in Neural Information Processing Systems 4*, pages 488–495, Denver, 1992. Morgan Kaufmann.

[5] T. Okuda, E. Tanaka, and K. Tamotsu. A method for the correction of garbled words based on the Levenshtein metric. *IEEE Transactions on Computers*, c-25(2):172–177, 1976.

[6] M. Schenkel and M. Jabri. Degraded printed document recognition using convolutional neural networks and hidden markov models. In *Proceedings of the ACNN*, Melbourne, 1997.

